# Committing Bandits

**Loc Bui**[*]
MS&E Department
Stanford University

**Ramesh Johari**[†]
MS&E Department
Stanford University

**Shie Mannor**[‡]
EE Department
Technion

## Abstract

We consider a multi-armed bandit problem where there are two phases. The first phase is an *experimentation* phase where the decision maker is free to explore multiple options. In the second phase the decision maker has to *commit* to one of the arms and stick with it. Cost is incurred during both phases with a higher cost during the experimentation phase. We analyze the regret in this setup, and both propose algorithms and provide upper and lower bounds that depend on the ratio of the duration of the experimentation phase to the duration of the commitment phase. Our analysis reveals that if given the choice, it is optimal to experiment $\Theta(\ln T)$ steps and then commit, where $T$ is the time horizon.

## 1 Introduction

In a range of applications, a dynamic decision making problem exhibits two distinctly different kinds of phases: *experimentation* and *commitment*. In the first phase, the decision maker explores multiple options, to determine which might be most suitable for the task at hand. However, eventually the decision maker must commit a choice, and use that decision for the duration of the problem horizon. A notable feature of these phases in the models we study is that costs are incurred during *both* phases; that is, experimentation is not carried out "offline," but rather is run "live" in the actual system.

For example, consider the design of a recommendation engine for an online retailer (such as Amazon). Experimentation amounts to testing different recommendation strategies on arriving customers. However, such testing is not carried out without consequences; the retailer might lose potential rewards if experimentation leads to suboptimal recommendations. Eventually, the recommendation engine must be stabilized (both from a software development standpoint and a customer expectation standpoint), and when this happens the retailer has effectively committed to one strategy moving forward. As another example, consider product design and delivery (e.g., tapeouts in semiconductor manufacturing, or major releases in software engineering). The process of experimentation during design entails costs to the producer, but eventually the experimentation must stop and the design must be committed. Another example is that of dating followed by marriage to hopefully, the best possible mate.

In this paper we consider a class of multi-armed bandit problems (which we call *committing bandit* problems) that mix these two features: the decision maker is allowed to try different arms in each period until commitment, at which point a final choice is made ("committed") and the chosen arm is used until the end of the horizon. Of course, models that investigate each phase in isolation are extensively studied. If the problem consists of only experimentation, then we have the classical multi-armed bandit problem, where the decision maker is interested in minimizing the expected total regret against the best arm [9, 2]. At the other extreme, several papers have studied the *pure*

---

[*]Email: `locbui@stanford.edu`
[†]Email: `ramesh.johari@stanford.edu`
[‡]Email: `shie@ee.technion.ac.il`

*exploration* or *budgeted learning* problem, where the goal is to output the best arm at the end of an experimentation phase [13, 6, 4]; no costs are incurred for experimentation, but after finite time a single decision must be chosen (see [12] for a review).

Formally, in a *committing bandit* problem, the decision maker can experiment without constraints for the first $N$ of $T$ periods, but must commit to a single decision for the last $T - N$ periods, where $T$ is the problem horizon. We first consider the *soft deadline* setting where the experimentation deadline $N$ can be chosen by the decision maker, but there is a cost incurred per experimentation period. We divide this setting into two regimes depending on how $N$ is chosen: the *non-adaptive* regime (Section 3) in which the decision maker has to choose $N$ before the algorithm begins running, and the *adaptive* regime (Section 4) in which $N$ can be chosen adaptively as the algorithm runs.

We obtain two main results for the soft deadline setting. First, in both regimes, we find that the best tradeoff between experimentation and commitment (in terms of expected regret performance) is essentially obtained by experimenting for $N = \Theta(\ln T)$ periods, and then committing to the empirical best action for the remaining $T - \Theta(\ln T)$ periods; this yields an expected average regret of $\Theta(\ln T/T)$. Second, and somewhat surprisingly, we find that if the algorithm has access to distributional information about the arms, then *adaptivity provides no additional benefit* (at least in terms of expected regret performance); however, as we observe via simulations, on a sample path basis adaptive algorithms can outperform nonadaptive algorithms due to the additional flexibility. Finally, we demonstrate that if the algorithm has no initial distributional information, adaptivity is beneficial: we demonstrate an adaptive algorithm that achieves $\Theta(\ln T/T)$ regret in this case.

We then study the *hard deadline* regime where the value of $N$ is given to the decision maker in advance (Section 5). This is a sensible assumption for problems where the decision maker cannot control how long the experimentation period is; for example, in the product design example above, the release date is often fixed well in advance, and the engineers are not generally free to alter it. We propose the **UCB-poly**($\delta$) algorithm for this setting, where the parameter $\delta \in (0, 1)$ reflects the tradeoff between experimentation and commitment. We show how to tune the algorithm to optimally choose $\delta$, based on the relative values of $N$ and $T$.

We mention in passing that the celebrated exploration-exploitation dilemma is also a major issue in our setup. During the first $N$ periods the tradeoff between exploration and exploitation exists bearing in mind that the last $T - N$ periods will be used solely for exploitation. This changes the standard setup so that exploration in the first $N$ periods becomes more important, as we shall see in our results.

## 2  The committing bandit problem

We first describe the setup of the classical stochastic multi-armed bandit problem, as it will serve as background for the committing bandit problem. In a stochastic multi-armed bandit problem, there are $K$ independent arms; each arm $i$, when pulled, returns a reward which is independently and identically drawn from a fixed Bernoulli distribution[1] with unknown parameter $\theta_i \in [0, 1]$. Let $I_t$ denote the index of the arm pulled at time $t$ ($I_t \in \{1, 2, \ldots, K\}$), and let $X_t$ denote the associated reward. Note that $\mathbb{E}[X_t] = \theta_{I_t}$. Also, we define the following notation:

$$\theta^* := \max_{1 \leq i \leq K} \theta_i, \quad i^* := \arg\max_{1 \leq i \leq K} \theta_i, \quad \Delta_i := \theta^* - \theta_i, \quad \Delta := \min_{i:\Delta_i > 0} \Delta_i.$$

An *allocation policy* is an algorithm that chooses the next arm to pull based on the sequence of past pulled arms and obtained rewards. The *cumulative regret* of an allocation policy $\mathcal{A}$ after time $n$ is:

$$R_n = \sum_{t=1}^{n} (X_t^* - X_t),$$

where $X_t^*$ is the reward that the algorithm would have received at time $t$ if it had pulled the optimal arm $i^*$. In other words, $R_n$ is the cumulative loss due to the fact that the allocation policy does not always pull the optimal arm. Let $T_i(n)$ be the number of times that arm $i$ is pulled up to time $n$.

Then:

$$\mathbb{E}[R_n] = \theta^* n - \sum_{i=1}^{K} \theta_i \mathbb{E}[T_i(n)] = \sum_{i \neq i^*} \Delta_i \mathbb{E}[T_i(n)].$$

The reader is referred to the supplementary material for some well-known allocation policies, e.g., **Unif** (Uniform allocation) and **UCB** (Upper Confidence Bound) [2].

A *recommendation policy* is an algorithm that tries to recommend the "best" arm based on the sequence of past pulled arms and obtained rewards. Suppose that after time $n$, a recommendation policy $\mathcal{R}$ recommends the arm $J_n$ as the "best" arm. Then the regret of recommendation policy $\mathcal{R}$ after time $n$, called the *simple regret* in [4], is defined as

$$r_n = \theta^* - \theta_{J_n} = \Delta_{J_n}.$$

The reader is also referred to the supplementary material for some natural recommendation policies, e.g., **EBA** (Empirical Best Arm) and **MPA** (Most Played Arm).

The *committing bandit* problem considered in this paper is a version of the stochastic multi-armed bandit problem in which the algorithm is forced to commit to only one arm after some period of time. More precisely, the problem setting is as follows. Let $T$ be the time horizon of the problem. From time 1 to some time $N$ ($N < T$), the algorithm can pull any arm in $\{1, 2, \ldots, K\}$. Then, from time $N + 1$ to the end of the horizon (time $T$), it must *commit* to pull only one arm. The first phase (time 1 to $N$) is called the *experimentation phase*, and the second phase (time $N + 1$ to $T$) is called the *commitment phase*. We refer to time $N$ as the *experimentation deadline*.

An algorithm for the committing bandit problem is a combination of an allocation and a recommendation policy. That is, the algorithm has to decide which arm to pull during the first $N$ slots, and then choose an arm to commit to during the remaining $T - N$ slots. Because we consider settings where the algorithm designer can choose the experimentation deadline, we also assume a cost is imposed during the experimentation phase; otherwise, it is never optimal to be forced to commit. In particular, we assume that the reward earned during the experimentation phase is reduced by a constant factor $\gamma \in [0, 1)$. Thus the expected regret $\mathbb{E}[\text{Reg}]$ of such an algorithm is the average regret across both phases, i.e.:

$$\mathbb{E}[\text{Reg}] = \frac{1}{T} \left( \sum_{t=1}^{T} \theta^* - \gamma \sum_{t=1}^{N} \mathbb{E}[\theta_{I_t}] - \sum_{t=N+1}^{T} \mathbb{E}[\theta_{J_N}] \right) = \gamma \frac{\mathbb{E}[R_N]}{T} + \frac{T - N}{T} \mathbb{E}[r_N] + (1 - \gamma) \frac{N \theta^*}{T}.$$

## 2.1 Committing bandit regimes

We focus on three distinct regimes, that differ in the level of control given to the algorithm designer in choosing the experimentation deadline.

**Regime 1: Soft experimentation deadline, non-adaptive.** In this regime, the value of $T$ is given to the algorithm. For a given value of $T$, the value of $N$ can be chosen freely between 1 and $T - 1$, but the choice must be made *before* the process begins.

**Regime 2: Soft experimentation deadline, adaptive.** The setting in this regime is the same as the previous one, except for the fact that the algorithm can choose the value of $N$ *adaptively* as outcomes of past pulls are observed.

**Regime 3: Hard experimentation deadline.** In this regime, *both* $N$ and $T$ are fixed and given to the algorithm. That is, the algorithm cannot control the experimentation deadline $N$. We are mainly interested in the asymptotic behavior of the algorithm when *both* $N$ and $T$ go to infinity.

## 2.2 Known lower-bounds

As mentioned in the Introduction section, the experimentation and commitment phases have each been extensively studied in isolation. In this subsection, we only summarize briefly the known lower bounds on cumulative regret and simple regret that will be used in the paper.

**Result 1** (Distribution-dependent lower bound on cumulative regret [9]). *For any allocation policy, and for any set of reward distributions such that their parameters $\theta_i$ are not all equal, there exists*

*an ordering of $(\theta_1, \ldots, \theta_K)$ such that*

$$\mathbb{E}[R_n] \geq \left( \sum_{i \neq i^*} \frac{\Delta_i}{D(p_i \| p^*)} + o(1) \right) \ln n,$$

*where $D(p_i \| p^*) = p_i \log \frac{p_i}{p^*} + p^* \log \frac{p^*}{p_i}$ is the Kullback-Leibler divergence between two Bernoulli reward distributions $p_i$ (of arm $i$) and $p^*$ (of the optimal arm), and $o(1) \to 0$ as $n \to \infty$.*

**Result 2** (Distribution-free lower bound on cumulative regret [13])**.** *There exist positive constants $c$ and $N_0$ such that for any allocation policy, there exists a set of Bernoulli reward distributions such that*

$$\mathbb{E}[R_n] \geq cK(\ln n - \ln K), \quad \forall n \geq N_0.$$

The difference between Result 1 and Result 2 is that the lower bound in the former depends on the parameters of reward distributions (hence, called *distribution-dependent*), while the lower bound in the latter does not (hence, called *distribution-free*). That means, in the latter case, the reward distributions can be chosen adversarially. Therefore, it should be clear that the *distribution-free* lower bound is always *higher* than the *distribution-dependent* lower bound.

**Result 3** (Distribution-dependent bound on simple regret [4])**.** *For any pair of allocation and recommendation policies, if the allocation policy can achieve an upper bound such that for all (Bernoulli) reward distributions $\theta_1, \ldots, \theta_K$, there exists a constant $C \geq 0$ with*

$$\mathbb{E}[R_n] \leq Cf(n),$$

*then for all sets of $K \geq 3$ Bernoulli reward distributions with parameters $\theta_i$ that are all distinct and all different from 1, there exists an ordering $(\theta_1, \ldots, \theta_K)$ such that*

$$\mathbb{E}[r_n] \geq \frac{\Delta}{2} e^{-Df(n)},$$

*where $D$ is a constant which can be calculated in closed form from $C$, and $\theta_1, \ldots, \theta_K$.*

*In particular, since $\mathbb{E}[R_n] \leq \theta^* n$ for any allocation policy, there exists a constant $\xi$ depending only on $\theta_1, \ldots, \theta_K$ such that $\mathbb{E}[r_n] \geq (\Delta/2) e^{-\xi n}$.*

**Result 4** (Distribution-free lower bound on simple regret [4])**.** *For any pair of allocation and recommendation policies, there exists a set of Bernoulli reward distributions such that $\mathbb{E}[r_n] \geq \frac{1}{20} \sqrt{\frac{K}{n}}$.*

In the subsequent sections we analyze each of the committing bandit regimes in detail; in particular, we provide constructive upper bounds and matching lower bounds on the regret in each regime. The detailed proofs of all the results in this paper are presented in the supplementary material.

## 3 Regime 1: Soft experimentation deadline, non-adaptive

In this regime, for a given value of $T$, the value of $N$ can be chosen freely between 1 and $T - 1$, but only before the algorithm begins pulling arms. Our main insight is that there exist matching upper and lower bounds of order $\Theta(\ln T/T)$; further, we propose an algorithm that can achieve this performance.

**Theorem 1.** *(1)* Distribution-dependent lower bound*: In Regime 1, for any algorithm, and any set of $K \geq 3$ Bernoulli reward distributions such that $\theta_i$ are all distinct and all different from 1, there exists an ordering $(\theta_1, \ldots, \theta_K)$ such that*

$$\mathbb{E}[\mathsf{Reg}] \geq \left( \max \left\{ \frac{(1-\gamma)\theta^*}{\xi}, \sum_{i \neq i^*} \frac{\Delta_i}{D(p_i \| p^*)} \right\} + o(1) \right) \frac{\ln T}{T},$$

*where $o(1) \to 0$ as $T \to \infty$, and $\xi$ is the constant discussed in Result 3.*

*(2)* Distribution-free lower bound*: Also, for any algorithm in Regime 1, there exists a set of Bernoulli reward distributions such that*

$$\mathbb{E}[\mathsf{Reg}] \geq cK \left( 1 - \frac{\ln K}{\ln T} \right) \frac{\ln T}{T},$$

*where $c$ is the constant in Result 2.*

We now show that the **Non-adaptive Unif-EBA** algorithm (Algorithm 1) achieves the matching upper bound, as stated in the following theorem.

---

**Algorithm 1** Non-adaptive Unif-EBA

---

**Input:** a set of arms $\{1, 2, \ldots, K\}, T, \Delta$
**repeat**
   Sample each arm in $\{1, 2, \ldots, K\}$ in the round robin fashion.
**until** each arm has been chosen $\lceil \ln T / \Delta^2 \rceil$ times.
Commit to the arm with *maximum empirical average reward* for the remaining periods.

---

**Theorem 2.** *For the* **Non-adaptive Unif-EBA** *algorithm (Algorithm 1),*

$$\mathbb{E}[\mathsf{Reg}] \ \leq \ \frac{K}{\Delta^2} \left( (1 - \gamma)\theta^* + \frac{\gamma}{K} \sum_{i \neq i^*} \Delta_i + \frac{2\Delta^2}{\ln T} \right) \frac{\ln T}{T}.$$

This matches the lower bounds in Theorem 1 to the correct order in $T$. Observe that in this regime, both distribution-dependent and distribution-free lower bounds have the same asymptotic order of $\ln T / T$. However, the preceding algorithm requires knowing the value of $\Delta$. If $\Delta$ is unknown, a low regret algorithm that matches the lower bound does not seem to be possible in this regime, because of the relative nature of the regret. An algorithm may be unable to choose an $N$ that explores sufficiently long when arms are difficult to distinguish, and yet commits quickly when arms are easy to distinguish.

## 4 Regime 2: Soft experimentation deadline, adaptive

The setting in this regime is the same as the previous one, except that the algorithm is not required to choose $N$ before it runs, i.e., $N$ can be chosen *adaptively*. Thus, in particular, it is possible for the algorithm to reject bad arms or to estimate $\Delta$ as it runs.

We first present the lower bounds on regret for any algorithm in this regime.

**Theorem 3.** *(1)* Distribution-dependent lower bound*: In Regime 2, for any algorithm, and any set of $K \geq 3$ Bernoulli reward distribution such that $\theta_i$ are all distinct and all different from $1$, there exists an ordering $(\theta_1, \ldots, \theta_K)$ such that*

$$\mathbb{E}[\mathsf{Reg}] \ \geq \ \left( \sum_{i \neq i^*} \frac{\Delta_i}{D(p_i \| p^*)} + o(1) \right) \frac{\ln T}{T},$$

*where $o(1) \to 0$ as $T \to \infty$.*

*(2)* Distribution-free lower bound*: Also, for any algorithm in Regime 2, there exists a set of Bernoulli reward distributions such that*

$$\mathbb{E}[\mathsf{Reg}] \ \geq \ cK \left( 1 - \frac{\ln K}{\ln T} \right) \frac{\ln T}{T},$$

*where $c$ is the constant in Result 2.*

Next, we derive several sequential algorithms with matching upper bounds on regret. The first algorithm is called **Sequential Elimination & Commitment 1 (SEC1)** (Algorithm 2); this algorithm requires the values of $\Delta$ and $\theta^*$.

**Theorem 4.** *For the* **SEC1** *algorithm (Algorithm 2),*

$$\mathbb{E}[\mathsf{Reg}] \ \leq \ \frac{K}{\Delta^2} \left( (1 - \gamma)\theta^* + \frac{\gamma}{K} \sum_{i \neq i^*} \Delta_i + b \right) \frac{\ln T}{T},$$

*where $b = \left( 2 + \frac{\Delta^2(K+2)}{(1 - e^{-\Delta^2/2})^2} \right) \frac{1}{\ln T} \to 0$ as $T \to \infty$.*

---

**Algorithm 2** Sequential Elimination & Commitment 1 (SEC1)

---

**Input:** A set of arms $\{1, 2, \ldots, K\}, T, \Delta, \theta^*$
**Initialization:** Set $m = 0, B_0 = \{1, 2, \ldots, K\}, \alpha = 1/\Delta^2, \epsilon_1 = 1/\Delta, \epsilon_2 = \Delta/2$.
**repeat**
    Sample each arm in $B_m$ once. Let $S_m^i$ be the total reward obtained from arm $i$ so far.
    Set $B_{m+1} = B_m, m = m + 1$.
    **for** $i \in B_m$ **do**
        **if** $m \leq \lceil \alpha \ln T \rceil$ **and** $|m\theta^* - S_m^i| > \epsilon_1 \ln T$ **then**
            Delete arm $i$ from $B_m$.
        **end if**
        **if** $m > \lceil \alpha \ln T \rceil$ **and** $|m\theta^* - S_m^i| > \epsilon_2 m$ **then**
            Delete arm $i$ from $B_m$.
        **end if**
    **end for**
**until** there is only one arm in $B_m$, then commit to that arm **or** the horizon $T$ is reached.

---

Observe that this algorithm matches the lower bounds in Theorem 3 to the correct order in $T$. We note that when $N$ can be chosen adaptively, both distribution-dependent and distribution-free lower bounds have the same asymptotic order of $\ln T/T$ as the ones in the non-adaptive regime. In the distribution-dependent case, therefore, we obtain the surprising conclusion that *adaptivity does not reduce the optimal expected regret*. Indeed, the regret bound of **SEC1** in Theorem 4 is exactly the same as for **Non-adaptive Unif-EBA** in Theorem 2. We conjecture that the constant $1/\Delta^2$ is actually the best achievable constant on expected regret.

What is the benefit of adaptivity then? As simulation results in Section 6 suggest, **SEC1** performs much better than **Non-adaptive Unif-EBA** in practice. The reason is rather intuitive: due to its adaptive nature, **SEC1** is able to eliminate poor arms much earlier than the $\lceil \ln T/\Delta^2 \rceil$ threshold, while **Non-adaptive Unif-EBA** has to wait until that point to make decisions.

**Remark 1.** *Although* **SEC1** *requires the value of* $\theta^*$*, that requirement can be relaxed as* $\theta^*$ *can be estimated by the maximum empirical average reward across arms. In fact, as we will see in the simulations (Section 6), another version of* **SEC1** *(called* **SEC2***) in which* $m\theta^*$ *is replaced by* $\max_{j \in B_m} S_m^j$ *achieves a nearly identical performance*.

Now, if the value of $\Delta$ is unknown, we have the following **Sequential Committing UCB (SC-UCB)** algorithm which is based on the improved UCB algorithm in [3]. The idea is to maintain an estimate of $\Delta$ and reduce it over time.

---

**Algorithm 3** Sequential Committing UCB (SC-UCB)

---

**Input:** A set of arms $\{1, 2, \ldots, K\}, T$
**Initialization:** Set $m = 0, \tilde{\Delta}_0 = 0, B_0 = \{1, 2, \ldots, K\}$.
**for** $m = 0, 1, 2, \ldots, \lfloor \log_2(T/e)/2 \rfloor$ **do**
    **if** $|B_m| > 1$ **then**
        Sample each arm in $B_m$ until each arm has been chosen $n_m = \left\lceil 2 \ln(T\tilde{\Delta}_m^2)/\tilde{\Delta}_m^2 \right\rceil$ times.
        Let $S_m^i$ be the total reward obtained from arm $i$ so far.
        Delete all arms $i$ from $B_m$ for which
$$\max_{j \in B_m} S_m^j - S_m^i > 2\sqrt{n_m \ln(T\tilde{\Delta}_m^2)/2}$$
        to obtain $B_{m+1}$.
        Set $\tilde{\Delta}_{m+1} = \tilde{\Delta}_m/2$.
    **else**
        Commit to the single arm in $B_m$.
    **end if**
**end for**
Commit to any arm in $B_m$.

---

**Theorem 5.** *For the **SC-UCB** algorithm (Algorithm 3),*

$$\mathbb{E}[\text{Reg}] \leq \sum_{i \neq i^*} \left( \frac{\gamma\Delta_i + (1-\gamma)\theta^*}{\Delta_i^2} \right) \frac{\ln(T\Delta_i^2)}{T} \left( 32 + \frac{\Delta_i^2 + 96}{\ln(T\Delta_i^2)} \right).$$

*This matches the lower bounds in Theorem 3 to the correct order in $T$.*

## 5 Regime 3: Hard experimentation deadline

We now investigate the third regime where, in contrast to the previous two, the experimentation deadline $N$ is *fixed* exogenously together with $T$. We consider the asymptotic behavior of regret as $T$ and $N$ approach infinity together. Note that since in this case the experimentation deadline is outside the algorithm designer's control, we set the cost of experimentation $\gamma = 1$ for this section.

Because both $T$ and $N$ are given, the main challenge in this context is choosing an algorithm that optimally balances the cumulative and simple regrets. We design and tune an algorithm that achieves this balance.

We know from Result 3 that for any pair of allocation and recommendation policies, if $\mathbb{E}[R_N] \leq C_1 f(N)$, then $\mathbb{E}[r_N] \geq (\Delta/2)e^{-Df(N)}$. In other words, given an allocation policy $\mathcal{A}$ that has a cumulative regret bound $C_1 f(N)$ (for some constant $C_1$), the best (distribution-dependent) upper bound that any recommendation policy can achieve is $C_2 e^{-C_3 f(N)}$ (for some constants $C_2$ and $C_3$). Assuming that there exists a recommendation policy $\mathcal{R}^{\mathcal{A}}$ that achieves such an upper bound, we have the following upper bound on regret when applying $[\mathcal{A}, \mathcal{R}^{\mathcal{A}}]$ to the committing bandit problem:

$$\mathbb{E}[\text{Reg}] \leq C_1 \frac{f(N)}{T} + \frac{T-N}{T} C_2 e^{-C_3 f(N)}. \tag{1}$$

One can clearly see the trade-off between experimentation and commitment in (1): the smaller the first term, the larger the second term, and vice versa. Note that $\ln(N) \leq f(N) \leq N$, and we have algorithms that give us only either one of the extremes (e.g., **Unif** has $f(N) = N$, while **UCB** [2] has $f(N) = \ln N$). On the other hand, it would be useful to have an algorithm that can balance between these two extremes. In particular, we focus on finding a pair of allocation and recommendation policies which can simultaneously achieve the allocation bound $C_1 N^\delta$ and the recommendation bound $C_2 e^{-C_3 N^\delta}$ where $0 < \delta < 1$.

Let us consider a modification of the UCB allocation policy called **UCB-poly**$(\delta)$ (for $0 < \delta < 1$), where for $t > K$, with $\hat{\theta}_{i,T_i(t-1)}$ be the empirical average of rewards from arm $i$ so far,

$$I_t = \arg\max_{1 \leq i \leq K} \left( \hat{\theta}_{i,T_i(t-1)} + \sqrt{\frac{2(t-1)^\delta}{T_i(t-1)}} \right).$$

Then we have the following result on the upper bound of its cumulative regret.

**Theorem 6.** *The cumulative regret of **UCB-poly**$(\delta)$ is upper-bounded by*

$$\mathbb{E}[R_n] \leq \left( \sum_{i:\Delta_i>0} \frac{8}{\Delta_i} + o(1) \right) n^\delta,$$

*where $o(1) \to 0$ as $n \to \infty$. Moreover, the simple regret for the pair **[UCB-poly**$(\delta)$**, EBA]** is upper-bounded by*

$$\mathbb{E}[r_n] \leq \left( 2\sum_{i \neq i^*} \Delta_i \right) e^{-\chi n^\delta},$$

*where $\chi = \min_i \frac{\sigma}{2}\Delta_i^2$.*

In the supplementary material (see Theorem 7 there) we show that in the limit, as $T$ and $N$ increase to infinity, the optimal value of $\delta$ can be chosen as $\lim_{N\to\infty} \ln(\ln(T(N) - N))/\ln N$ if that limit exists. In particular, if $T(N)$ is super-exponential in $N$ we get an optimal $\delta$ of 1 representing pure exploration in the experimentation phase. If $T(N)$ is sub-exponential we get an optimal $\delta$ of 0 representing a standard UCB during the experimentation phase. If $T(N)$ is exponential we obtain $\delta$ in between.

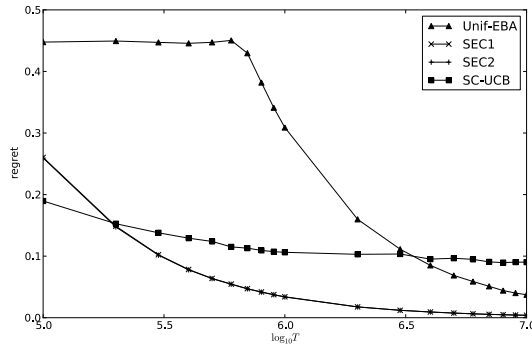

Figure 1: Numerical performances where $K = 20, \gamma = 0.75$, and $\Delta = 0.02$

## 6 Simulations

In this section, we present numerical results on the performance of **Non-adaptive Unif-EBA**, **SEC1**, **SEC2**, and **SC-UCB** algorithms. (Recall that the **SEC2** algorithm is a version of **SEC1** in which $m\theta^*$ is replaced by $\max_{j \in B_m} S_m^j$, as discussed in Remark 1). The simulation setting includes $K$ arms with Bernoulli reward distributions, the time horizon $T$, and the values of $\gamma$ and $\Delta$. The arm configurations are generated as follows. For each experiment, $\theta^*$ is generated independently and uniformly in the $[0.5, 1]$ interval, and the second best arm reward is set as $\theta_2^* = \theta^* - \Delta$. These two values are then assigned to two randomly chosen arms, and the rest of arm rewards are generated independently and uniformly in $[0, \theta_2^*]$.

Figure 1 shows the regrets of the above algorithms for various values of $T$ (in logarithmic scale) with parameters $K = 20, \gamma = 0.75$, and $\Delta = 0.02$ (we omitted error bars because the variation was small). Observe that the performances of **SEC1** and **SEC2** are nearly identical, which suggests that the requirement of knowing $\theta^*$ in **SEC1** can be relaxed (see Remark 1). Moreover, **SEC1** (or equivalently, **SEC2**) performs much better than **Non-adaptive Unif-EBA** due to its adaptive nature (see the discussion before Remark 1). Particularly, the performance of **Non-adaptive Unif-EBA** is quite poor when the experimentation deadline is roughly equal to $T$, since the algorithm does not commit before the experimentation deadline. Finally, **SC-UCB** does not perform as well as the others when $T$ is large, but this algorithm does not need to know $\Delta$, and thus suffers a performance loss due to the additional effort required to estimate $\Delta$.

Additional simulation results can be found in the supplementary material.

## 7 Extensions and future directions

Our work is a first step in the study of the committing bandit setup. There are several extensions that call for future research which we outline below.

First, an extension of the basic committing bandits setup to the case of contextual bandits [10, 11] is natural. In this setup before choosing an arm an additional "context" is provided to the decision maker. The problem is to choose a decision rule from a given class that prescribes what arm to choose for every context. This setup is more realistic when the decision maker has to commit to such a rule after some exploration time. Second, models with many arms (structured as in [8, 5]) or even infinitely arms (as in [1, 7, 14]) are of interest here as they may lead to different regimes and results here. Third, our models assumed that the commitment time is either predetermined or according to the decision maker's will. There are other models of interest such as the case where some stochastic process determines the commitment time.

Finally, a situation where the exploration and commitment phases alternate (randomly or according to a given schedule or at a cost) is of practical interest. This can represent the situation where there are a few releases of a product where exploration can be done until the time of the release, when the product is "frozen" until a new exploration period followed by a new release.

## Footnotes

[1]We assume Bernoulli distributions throughout the paper. Our results hold with minor modification for any distribution with bounded support.

## References

[1] R. Agrawal. The continuum-armed bandit problem. *SIAM Journal on Control and Optimization*, 33(6):1926–1951, 1995.

[2] P. Auer, N. Cesa-Bianchi, and P. Fischer. Finite-time analysis of the multi-armed bandit problem. *Machine Learning Journal*, 47(2-3):235–256, 2002.

[3] P. Auer and R. Ortner. UCB revisited: Improved regret bounds for the stochastic multi-armed bandit problem. *Periodica Mathematica Hungarica*, 61(1-2):55–65, 2010.

[4] S. Bubeck, R. Munos, and G. Stoltz. Pure exploration in finitely-armed and continuous-armed bandits. *Theoretical Computer Science*, 412(19):1832–1852, 2011.

[5] P. A. Coquelin and R. Munos. Bandit algorithms for tree search. *CoRR*, abs/cs/0703062, 2007.

[6] E. Even-Dar, S. Mannor, and Y. Mansour. Action elimination and stopping conditions for the multi-armed bandit and reinforcement learning problems. *Journal of Machine Learning Research*, 7:1079–1105, 2006.

[7] R. Kleinberg, A. Slivkins, and E. Upfal. Multi-armed bandits in metric spaces. In *STOC*, pages 681–690, 2008.

[8] L. Kocsis and C. Szepesvári. Bandit based Monte-Carlo planning. In *ECML*, pages 282–293, 2006.

[9] T. L. Lai and H. Robbins. Asymptotically efficient adaptive allocation rules. *Advances in Applied Mathematics*, 6:4–22, 1985.

[10] J. Langford and T. Zhang. The epoch-greedy algorithm for contextual multi-armed bandits. In *Advances in Neural Information Processing (NIPS)*, 2008.

[11] L. Li, W. Chu, J. Langford, and R.E. Schapire. A contextual-bandit approach to personalized news article recommendation. In *Proceedings of the 19th International Conference on World Wide Web*, pages 661–670, 2010.

[12] S. Mannor. $k$-armed bandit. In *Encyclopedia of Machine Learning*, pages 561–563. 2010.

[13] S. Mannor and J. Tsitsiklis. The sample complexity of exploration in the multi-armed bandit problem. *Journal of Machine Learning Research*, 5:623–648, 2004.

[14] P. Rusmevichientong and J. Tsitsiklis. Linearly parameterized bandits. *Mathematics of Operations Research*, 35(2):395–411, 2010.

